# Learning and Tracking Cyclic Human Motion

**D. Ormoneit**
Dept. of Computer Science
Stanford University
Stanford, CA 94305
ormoneit@cs.stanford.edu

**H. Sidenbladh**
Royal Institute of Technology (KTH),
CVAP/NADA,
S–100 44 Stockholm, Sweden
hedvig@nada.kth.se

**M. J. Black**
Dept. of Computer Science
Brown University, Box 1910
Providence, RI 02912
black@cs.brown.edu

**T. Hastie**
Dept. of Statistics
Stanford University
Stanford, CA 94305
hastie@stat.stanford.edu

## Abstract

We present methods for learning and tracking human motion in video. We estimate a statistical model of typical activities from a large set of 3D periodic human motion data by segmenting these data automatically into "cycles". Then the mean and the principal components of the cycles are computed using a new algorithm that accounts for missing information and enforces smooth transitions between cycles. The learned temporal model provides a prior probability distribution over human motions that can be used in a Bayesian framework for tracking human subjects in complex monocular video sequences and recovering their 3D motion.

## 1 Introduction

The modeling and tracking of human motion in video is important for problems as varied as animation, video database search, sports medicine, and human-computer interaction. Technically, the human body can be approximated by a collection of articulated limbs and its motion can be thought of as a collection of time-series describing the joint angles as they evolve over time. A key challenge in modeling these joint angles involves decomposing the time-series into suitable temporal primitives. For example, in the case of repetitive human motion such as walking, motion sequences decompose naturally into a sequence of "motion cycles". In this work, we present a new set of tools that carry out this segmentation automatically using the signal-to-noise ratio of the data in an aligned reference domain. This procedure allows us to use the mean and the principal components of the individual cycles in the reference domain as a statistical model. Technical difficulties include missing information in the motion time-series (resulting from occlusions) and the necessity of enforcing smooth transitions between different cycles. To deal with these problems,

we develop a new iterative method for functional Principal Component Analysis (PCA). The learned temporal model provides a prior probability distribution over human motions that can be used in a Bayesian framework for tracking. The details of this tracking framework are described in [7] and are briefly summarized here. Specifically, the posterior distribution of the unknown motion parameters is represented using a discrete set of samples and is propagated over time using particle filtering [3, 7]. Here the prior distribution based on the PCA representation improves the efficiency of the particle filter by constraining the samples to the most likely regions of the parameter space. The resulting algorithm is able to track human subjects in monocular video sequences and to recover their 3D motion under changes in their pose and against complex unknown backgrounds.

Previous work on modeling human motion has focused on the recognition of activities using Hidden Markov Models (HMM's), linear dynamical models, or vector quantization (see [7, 5] for a summary of related work). These approaches typically provide a coarse approximation to the underlying motion. Alternatively, explicit temporal curves corresponding to joint motion may be derived from biometric studies or learned from 3D motion-capture data. In previous work on principal component analysis of motion data, the 3D motion curves corresponding to particular activities had typically to be hand-segmented and aligned [1, 7, 8]. By contrast, this paper details an automated method for segmenting the data into individual activities, aligning activities from different examples, modeling the statistical variation in the data, dealing with missing data, enforcing smooth transitions between cycles, and deriving a probabilistic model suitable for a Bayesian interpretation. We focus here on cyclic motions which are a particularly simple but important class of human activities [6]. While Bayesian methods for tracking 3D human motion have been suggested previously [2, 4], the prior information obtained from the functional PCA proves particularly effective for determining a low-dimensional representation of the possible human body positions [8, 7].

## 2 Learning

Training data is provided by a commercial motion capture system describes the evolution of $m = 19$ relative joint angles over a period of about 500 to 5000 frames. We refer to the resulting multivariate time-series as a "motion sequence" and we use the notation $Z_i(t) \equiv \{z_{a,i}(t) | a = 1, \ldots, m\}$ for $t = 1, \ldots, T_i$ to denote the angle measurements. Here $T_i$ denotes the length of sequence $i$ and $a = 1, \ldots, m$ is the index for the individual angles. Altogether, there are $n = 20$ motion sequences in our training set. Note that missing observations occur frequently as body markers are often occluded during motion capture. An associated set $I_{a,i} \equiv \{t \in \{1, \ldots, T_i\} \, | \, z_{a,i}(t) \text{ is not missing}\}$ indicates the positions of valid data.

### 2.1 Sequence Alignment

Periodic motion is composed of repetitive "cycles" which constitute a natural unit of statistical modeling and which must be identified in the training data prior to building a model. To avoid error-prone manual segmentation we present alignment procedures that segment the data automatically by separately estimating the cycle length and a relative offset parameter for each sequence. The cycle length is computed by searching for the value $p$ that maximizes the "signal-to-noise ratio":

$$stn\_ratio_i(p) \equiv \sum_a \frac{signal_{i,a}(p)}{noise_{i,a}(p)}, \tag{1}$$

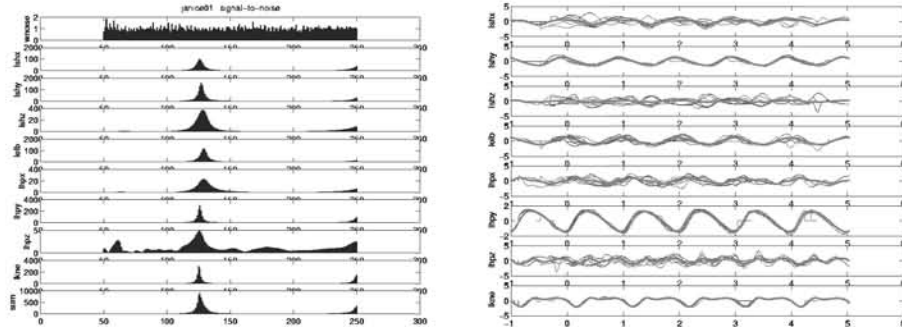

Figure 1: *Left:* Signal-to-noise ratio of a representative set of angles as a function of the candidate period length. *Right:* Aligned representation of eight walking sequences.

where $noise_{i,a}(p)$ is the variation in the data that is not explained by the mean cycle, $\bar{z}$, and $signal_{i,a}(p)$ measures the signal intensity.[1] In Figure 1 we show the individual signal-to-noise ratios for a subset of the angles as well as the accumulated signal-to-noise ratio as functions of $p$ in the range $\{50, 51, \ldots, 250\}$. Note the peak of these values around the optimal cycle length $p = 126$. Note also that the signal-to-noise ratio of the white noise series in the first row is approximately constant, warranting the unbiasedness of our approach.

Next, we estimate the offset parameters, $o$, to align multiple motion sequences in a common domain. Specifically, we choose $o(1), o(2), \ldots, o(n)$ so that the shifted motion sequences minimize the deviation from a common prototype model by analogy to the signal-to-noise-criterion (1). An exhaustive search for the optimal offset combination is computationally infeasible. Instead, we suggest the following iterative procedure: We initialize the offset values to zero in Step 1, and we define a *reference signal* $r_a$ in Step 2 so as to minimize the deviation with respect to the aligned data. This reference signal is a periodically constrained regression spline that ensures smooth transitions at the boundaries between cycles. Next, we choose the offsets of all sequences so that they minimize the prediction error with respect to the reference signal (Step 3). By contrast to the exhaustive search, this operation requires $O\left(\sum_{i=1}^{n} p(i)\right)$ comparisons. Because the solution of the first iteration may be suboptimal, we construct an improved reference signal using the current offset estimates, and use this signal in turn to improve the offset estimates. Repeating these steps, we obtain an iterative optimization algorithm that is terminated if the improvement falls below a given threshold. Because Steps 2 and 3 both decrease the prediction error, so that the algorithm converges monotonically. Figure 1 (right) shows eight joint angles of a walking motion, aligned using this procedure.

## 2.2 Functional PCA

The above alignment procedures segment the training data into a collection of cycle-data called "slices". Next, we compute the principal components of these slices, which can be interpreted as the major sources of variation in the data. The algorithm is as follows

Our algorithm addresses several difficulties. First, even though the individual motion sequences are aligned in Figure 1, they are still sampled at different frequencies in the reference domain due to the different alignment parameters. This problem is accommodated in Step 1c by resampling after computing a functional estimate in continuous time in Step 1b. Second, missing data in the design matrix $X$ means we cannot simply use the Singular Value Decomposition (SVD) of $X^{(1)}$ to obtain the principal components. Instead we use an iterative approximation scheme [9] in which we alternate between an SVD step (4 through 7) and a data imputation step (8), where each update is designed so as to decrease the matrix distance between $X$ and its reconstruction, $X^{(4)}$. Finally, we need to ensure that the mean estimates and the principal components produce a smooth motion when recombined into a new sequence. Specifically, the approximation of an individual cycle must be periodic in the sense that its first two derivatives match at the left and the right endpoint. This is achieved by translating the cycles into a Fourier domain and by truncating high-frequency coefficients (Step 4). Then we compute the SVD in the Fourier domain in Step 5, and we reconstruct the design matrix using a rank-$q$ approximation in Steps 6 and 7, respectively. In Step 8 we use the reconstructed values as improved estimates for the missing data in $X$, and then we repeat Steps 4 through 7 using these improved estimates. This iterative process is continued until the performance improvement falls below a given threshold. As its output, the algorithm generates the imputed design matrix, $X$, as well as its principal components.

## 3  Bayesian Tracking

In tracking, our goal is to calculate the posterior probability distribution over 3D human poses given a sequence of image measurements, $\vec{I}_t$. The high dimensionality of the body model makes this calculation computationally demanding. Hence, we use the learned model above to constrain the body motions to valid walking motions. Towards that end, we use the SVD of $X^{(2)}$ to formulate a prior distribution for Bayesian tracking.

Formally, let $\theta(t) \equiv (\theta_a(t)|a = 1, \ldots, m)$ be a random vector of the relative joint angles at time $t$; i.e., the value of a motion sequence, $Z_i(t)$, at time $t$ is interpreted as the $i$-th realization of $\theta(t)$. Then $\theta(t)$ can be written in the form

$$\theta(t) = \tilde{\mu}(\psi_t) + \sum_{k=1}^{q} c_{t,k} v_k(\psi_t), \tag{2}$$

where $v_k$ is the Fourier inverse of the $k$-th column of $V$, rearranged as an $\mathcal{T} \times m$-matrix; similarly, $\tilde{\mu}$ denotes the rearranged mean vector $\mu$. $v_k(\psi)$ is the $\psi$-th column of $v_k$, and the $c_{t,k}$ are time-varying coefficients. $\psi_t \in \{0, \mathcal{T} - 1\}$ maps absolute time onto relative cycle positions or phases, and $\rho_t$ denotes the speed of the motion such that $\psi_{t+1} = (\psi_t + \rho_t) \mod \mathcal{T}$. Given representation (2), body positions are characterized entirely by the low-dimensional state-vector $\phi_t = (\mathbf{c}_t, \psi_t, \rho_t, \boldsymbol{\tau}_t^g, \boldsymbol{\theta}_t^g)'$, where $\mathbf{c}_t = (c_{t,1}, \ldots, c_{t,q})$ and where $\boldsymbol{\tau}_t^g$ and $\boldsymbol{\theta}_t^g$ represent the global 3D translation and rotation of the torso, respectively. Hence we the problem is to calculate the posterior distribution of $\phi_t$ given images up to time $t$. Due to the Markovian structure underlying $\phi_t$, this posterior distribution is given recursively by:

$$p(\phi_t \,|\, \vec{\mathbf{I}}_t) \propto p(\mathbf{I}_t \,|\, \phi_t) \int p(\phi_t \,|\, \phi_{t-1}) p(\phi_{t-1} \,|\, \vec{\mathbf{I}}_{t-1}) \, d\phi_{t-1} \,. \tag{3}$$

Here $p(\mathbf{I}_t \,|\, \phi_t)$ is the likelihood of observing the image $\mathbf{I}_t$ given the parameters and $p(\phi_{t-1} \,|\, \vec{\mathbf{I}}_{t-1})$ is the posterior probability from the previous instant. $p(\phi_t \,|\, \phi_{t-1})$ is a temporal prior probability distribution that encodes how the parameters $\phi_t$ change over time. The elements of the Bayesian approach are summarized below; for details the reader is referred to [7].

**Generative Image Model.** Let $M(\mathbf{I}_t, \phi_t)$ be a function that takes image texture at time $t$ and, given the model parameters, maps it onto the surfaces of the 3D model using the camera model. Similarly, let $M^{-1}(\cdot)$ take a 3D model and project its texture back into the image. Given these functions, the generative model of images at time $t + 1$ can be viewed as a mapping from the image at time $t$ to images at time $t + 1$:

$$\mathbf{I}_{t+1} = M^{-1}(M(\mathbf{I}_t, \phi_t), \phi_{t+1}) + \eta, \quad \eta \sim G(0, \sigma),$$

where $G(0, \sigma)$ denotes a Gaussian distribution with zero mean and standard deviation $\sigma$ and $\sigma$ depends on the viewing angle of the limb with respect to the camera and increases as the limb is viewed more obliquely (see [7] for details).

**Temporal Prior.** The temporal prior, $p(\phi_t \,|\, \phi_{t-1})$, models how the parameters describing the body configuration are expected to vary over time. The individual components of $\phi$, $(\mathbf{c}_t, \psi_t, \rho_t, \boldsymbol{\tau}_t^g, \boldsymbol{\theta}_t^g)$, are assumed to follow a random walk with Gaussian increments.

**Likelihood Model.** Given the generative model above we can compare the image at time $t - 1$ to the image $\mathbf{I}_t$ at $t$. Specifically, we compute this likelihood term separately for each limb. To avoid numerical integration over image regions, we generate $n_s$ pixel locations stochastically. Denoting the $i$th sample for limb $j$ as $\mathbf{x}_{j,i}$, we obtain the following measure of discrepancy:

$$E \equiv \sum_{i=1}^{n} (\mathbf{I}_t(\mathbf{x}_{j,i}) - M^{-1}(M(\mathbf{I}_{t-1}, \phi_{t-1}), \phi_t)(\mathbf{x}_{j,i}))^2. \tag{4}$$

As an approximate likelihood term we use

$$p(\mathbf{I}_t \,|\, \phi_t) = \prod_j \frac{q(\alpha_j)}{\sqrt{2\pi}\sigma(\alpha_j)} \exp(-E/(2\sigma(\alpha_j)^2 n_s)) + (1 - q(\alpha_j))p_{occluded}, \tag{5}$$

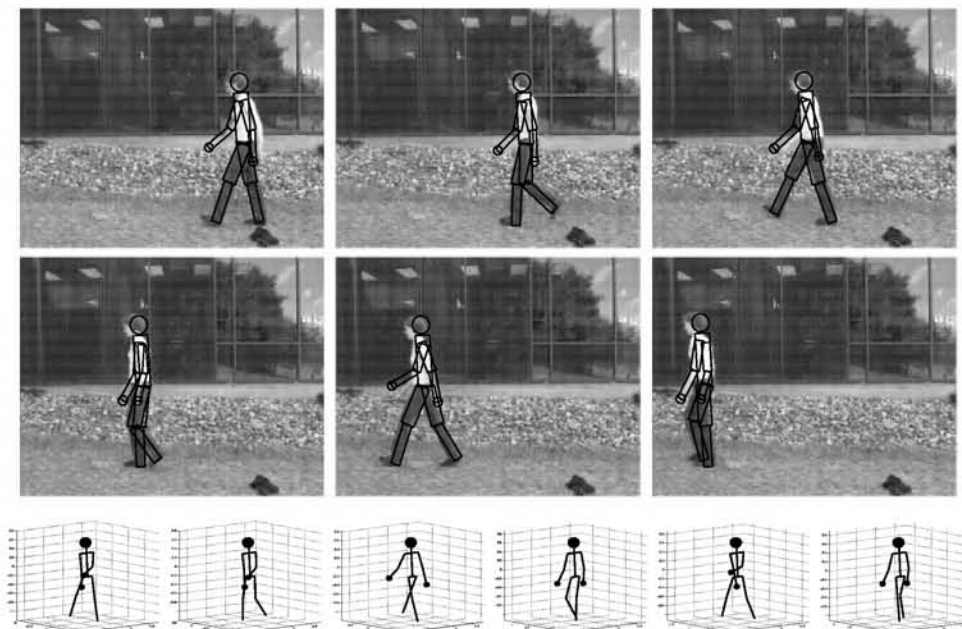

Figure 2: Tracking of person walking, 10000 samples. Upper rows: frames 0, 10, 20, 30, 40, 50 with the projection of the expected model configuration overlaid. Lower row: expected 3D configuration in the same frames.

where $p_{occluded}$ is a constant probability that a limb is occluded, $\alpha_j$ is the angle between the limb $j$ principal axis and the image plane of the camera, $\sigma(\alpha_j)$ is a function that increases with narrow viewing angles, and $q(\alpha_j) = cos(\alpha_j)$ if limb $j$ is non-occluded, or 0 if limb $j$ is occluded.

**Partical Filter.** As it is typical for tracking problems, the posterior distribution may well be multi-modal due to the nonlinearity of the likelihood function. Hence, we use a particle filter for inference where the posterior is represented as a weighted set of state samples, $\phi_i$, which are propagated in time. In detail, we use $N_s \approx 10^4$ particles in our experiments. Details of this algorithm can be found in [3, 7].

## 4 Experiment

To illustrate the method we show an example of tracking a walking person in a cluttered scene in Figure 2. The 3D motion is recovered from a monocular sequence using only the motion between frames. To visualize the posterior distribution we display the projection of the 3D model corresponding to the expected value of the model parameters: $\frac{1}{N_s} \sum_{i=1}^{N_s} p_i \phi_i$ where $p_i$ is the likelihood of sample $\phi_i$. All parameters were initialized manually with a Gaussian prior at time $t = 0$. The learned model is able to generalize to the subject in the sequence who was not part of the training set.

## 5 Conclusions

We described an automated method for learning periodic human motions from training data using statistical methods for detecting the length of the periods in the

data, segmenting it into cycles, and optimally aligning the cycles. We also presented a PCA method for building a statistical eigen-model of the motion curves that copes with missing data and enforces smoothness between the beginning and ending of a motion cycle. The learned eigen-curves are used as a prior probability distribution in a Bayesian tracking framework. Tracking in monocular image sequences was performed using a particle filtering technique and results were shown for a cluttered image sequence.

**Acknowledgements.** We thank M. Gleicher for generously providing the 3D motion-capture data and M. Kamvysselis and D. Fleet for many discussions on human motion and Bayesian estimation. Portions of this work were supported by the **Xerox Corporation** and we gratefully acknowledge their support.

## Footnotes

[1]The mean cycle is obtained by "folding" the original sequence into the domain $\{1, \ldots, p\}$. For brevity, we don't provide formal definitions here; see [5].

# References

[1] A. Bobick and J. Davis. An appearance-based representation of action. *ICPR*, 1996.

[2] T-J. Cham and J. Rehg. A multiple hypothesis approach to figure tracking. *CVPR*, pp. 239–245, 1999.

[3] M. Isard and A. Blake. Contour tracking by stochastic propagation of conditional density. *ECCV*, pp. 343–356, 1996.

[4] M. E. Leventon and W. T. Freeman. Bayesian estimation of 3-d human motion from an image sequence. Tech. Report TR–98–06, Mitsubishi Electric Research Lab, 1998.

[5] D. Ormoneit, H. Sidenbladh, M. Black, T. Hastie, Learning and tracking human motion using functional analysis, submitted: *IEEE Workshop on Human Modeling, Analysis and Synthesis*, 2000.

[6] S.M. Seitz and C.R. Dyer. Affine invariant detection of periodic motion. *CVPR*, pp. 970–975, 1994.

[7] H. Sidenbladh, M. J. Black, and D. J. Fleet. Stochastic tracking of 3D human figures using 2D image motion. to appear, *ECCV-2000*, Dublin Ireland.

[8] Y. Yacoob and M. Black. Parameterized modeling and recognition of activities in temporal surfaces. *CVIU*, 73(2):232–247, 1999.

[9] G. Sherlock, M. Eisen, O. Alter, D. Botstein, P. Brown, T. Hastie, and R. Tibshirani. "Imputing missing data for gene expression arrays," 2000, Working Paper, Department of Statistics, Stanford University.
